# Graph-Based Visual Saliency

**Jonathan Harel,   Christof Koch ,   Pietro Perona**
California Institute of Technology
Pasadena, CA 91125
{harel,koch}@klab.caltech.edu, perona@vision.caltech.edu

## Abstract

A new bottom-up visual saliency model, Graph-Based Visual Saliency (GBVS), is proposed. It consists of two steps: first forming activation maps on certain feature channels, and then normalizing them in a way which highlights conspicuity and admits combination with other maps. The model is simple, and biologically plausible insofar as it is naturally parallelized. This model powerfully predicts human fixations on 749 variations of 108 natural images, achieving 98% of the ROC area of a human-based control, whereas the classical algorithms of Itti & Koch ([2], [3], [4]) achieve only 84%.

## 1   Introduction

Most vertebrates, including humans, can move their eyes. They use this ability to sample in detail the most relevant features of a scene, while spending only limited processing resources elsewhere. The ability to predict, given an image (or video), where a human might fixate in a fixed-time free-viewing scenario has long been of interest in the vision community. Besides the purely scientific goal of understanding this remarkable behavior of humans, and animals in general, to consistently fixate on "important" information, there is tremendous engineering application, e.g. in compression and recognition [13]. The standard approaches (e.g., [2], [9]) are based on biologically motivated feature selection, followed by center-surround operations which highlight local gradients, and finally a combination step leading to a "master map". Recently, Bruce [5] and others [4] have hypothesized that fundamental quantities such as "self-information" and "surprise" are at the heart of saliency/attention. However, ultimately, Bruce computes a function which is additive in feature maps, with the main contribution materializing as a method of operating on a feature map in such a way to get an activation, or saliency, map. Itti and Baldi define "surprise" in general, but ultimately compute a saliency map in the classical [2] sense for each of a number of feature channels, then operate on these maps using another function aimed at highlighting local variation. By organizing the topology of these varied approaches, we can compare them more rigorously: i.e., not just end-to-end, but also piecewise, removing some uncertainty about the origin of observed performance differences.

Thus, the leading models of visual saliency may be organized into the these three stages:

(s1) **extraction**: extract feature vectors at locations over the image plane

(s2) **activation**: form an "activation map" (or maps) using the feature vectors

(s3) **normalization/combination**: normalize the activation map (or maps, followed by a combination of the maps into a single map)

In this light, [5] is a contribution to step (s2), whereas [4] is a contribution to step (s3). In the classic algorithms, step (s1) is done using biologically inspired filters, step (s2) is accomplished by subtracting feature maps at different scales (henceforth, "c-s" for "center" - "surround"), and step (s3) is accomplished in one of three ways: 1.  a normalization scheme based on local maxima

[2] ( "max-ave"),  2. an iterative scheme based on convolution with a difference-of-gaussians filter ("DoG"), and 3.  a nonlinear interactions ("NL") approach which divides local feature values by weighted averages of surrounding values in a way that is modelled to fit psychophysics data [11].

We take a different approach, exploiting the computational power, topographical structure, and parallel nature of graph algorithms to achieve natural and efficient saliency computations.  We define Markov chains over various image maps, and treat the equilibrium distribution over map locations as activation and saliency values. This idea is not completely new: Brockmann and Geisel [8] suggest that scanpaths might be predicted by properly defined Levy flights over saliency fields, and more recently Boccignone and Ferraro [7] do the same.  Importantly, they assume that a saliency map is *already available*, and offer an alternative to the winner-takes-all approach of mapping this object to a set of fixation locations. In an unpublished pre-print, L.F. Costa [6] notes similar ideas, however offers only sketchy details on how to apply this to real images, and in fact includes no experiments involving fixations.  Here, we take a unified approach to steps (s2) and (s3) of saliency computation, by using dissimilarity and saliency to define edge weights on graphs which are interpreted as Markov chains. Unlike previous authors, we do not attempt to connect features only to those which are somehow similar. We also directly compare our method to others, using power to predict human fixations as a performance metric.

The contributions of this paper are as follows:

(1) A complete bottom-up saliency model based on graph computations, GBVS, including a framework for "activation" and "normalization/combination".

(2) A comparison of GBVS against existing benchmarks on a data set of grayscale images of natural environments (viz., foliage) with the eye-movement fixation data of seven human subjects, from a recent study by Einhäuser et. al. [1].

## 2   The Proposed Method: Graph-Based Saliency (GBVS)

Given an image $I$, we wish to ultimately highlight a handful of 'significant' locations where the image is 'informative' according to some criterion, e.g. human fixation.  As previously explained, this process is conditioned on first computing feature maps (s1), e.g. by linear filtering followed by some elementary nonlinearity [15].  "Activation" (s2), "normalization and combination" (s3) steps follow as described below.

### 2.1   Forming an Activation Map (s2)

Suppose we are given a feature map[1] $M : [n]^2 \rightarrow \mathbb{R}$. Our goal is to compute an activation map $A : [n]^2 \rightarrow \mathbb{R}$, such that, intuitively, locations $(i,j) \in [n]^2$ where $I$, or as a proxy, $M(i,j)$, is somehow unusual in its neighborhood will correspond to high values of *activation* $A$.

#### 2.1.1   Existing Schemes

Of course "unusual" does not constrain us sufficiently, and so one can choose several operating definitions.  "Improbable" would lead one to the formulation of Bruce [5], where a histogram of $M(i,j)$ values is computed in some region around $(i,j)$, subsequently normalized and treated as a probability distribution, so that $A(i,j) = -\log(p(i,j))$ is clearly defined with $p(i,j) = \Pr\{M(i,j)|neighborhood\}$. Another approach compares local "center" distributions to broader "surround" distributions and calls the Kullback-Leibler tension between the two "surprise" [4].

### 2.1.2 A Markovian Approach

We propose a more organic (see below) approach. Let us define the dissimilarity of $M(i, j)$ and $M(p, q)$ as

$$d((i, j)\|(p, q)) \triangleq \left| \log \frac{M(i, j)}{M(p, q)} \right|.$$

This is a natural definition of dissimilarity: simply the distance between one and the ratio of two quantities, measured on a logarithmic scale. For some of our experiments, we use $|M(i, j) - M(p, q)|$ instead, and we have found that both work well. Consider now the fully-connected directed graph $G_A$, obtained by connecting every node of the lattice $M$, labelled with two indices $(i, j) \in [n]^2$, with all other $n - 1$ nodes. The directed edge from node $(i, j)$ to node $(p, q)$ will be assigned a weight

$$
\begin{aligned}
w_1((i, j), (p, q)) &\triangleq d((i, j)\|(p, q)) \cdot F(i - p, j - q), \text{ where} \\
F(a, b) &\triangleq \exp\left( -\frac{a^2 + b^2}{2\sigma^2} \right).
\end{aligned}
$$

$\sigma$ is a free parameter of our algorithm[2]. Thus, the weight of the edge from node $(i, j)$ to node $(p, q)$ is proportional to their dissimilarity and to their closeness in the domain of $M$. Note that the edge in the opposite direction has exactly the same weight. We may now define a Markov chain on $G_A$ by normalizing the weights of the outbound edges of each node to 1, and drawing an equivalence between nodes & states, and edges weights & transition probabilities . The equilibrium distribution of this chain, reflecting the fraction of time a random walker would spend at each node/state if he were to walk forever, would naturally accumulate mass at nodes that have high dissimilarity with their surrounding nodes, since transitions into such subgraphs is likely, and unlikely if nodes have similar $M$ values. The result is an activation measure which is derived from pairwise contrast.

We call this approach "organic" because, biologically, individual "nodes" (neurons) exist in a connected, retinotopically organized, network (the visual cortex), and communicate with each other (synaptic firing) in a way which gives rise to emergent behavior, including fast decisions about which areas of a scene require additional processing. Similarly, our approach exposes connected (via $F$) regions of dissimilarity (via $w$), in a way which can in principle be computed in a completely parallel fashion. Computations can be carried out independently at each node: in a synchronous environment, at each time step, each node simply sums incoming mass, then passes along measured partitions of this mass to its neighbors according to outbound edge weights. The same simple process happening at all nodes simultaneously gives rise to an equilibrium distribution of mass.

**Technical Notes**   The equilibrium distribution of this chain exists and is unique because the chain is ergodic, a property which emerges from the fact that our underlying graph $G_A$ is by construction strongly connected. In practice, the equilibrium distribution is computed using repeated multiplication of the Markov matrix with an initially uniform vector. The process yields the principal eigenvector of the matrix. The computational complexity is thus $O(n^4 K)$ where $K \ll n^2$ is some small number of iterations required to meet equilibrium[3].

### 2.2 "Normalizing" an Activation Map (s3)

The aim of the "normalization" step of the algorithm is much less clear than that of the activation step. It is, however, critical and a rich area of study. Earlier, three separate approaches were mentioned as existing benchmarks, and also the recent work of Itti on surprise [4] comes into the saliency computation at this stage of the process (although it can also be applied to s2 as mentioned above). We shall state the goal of this step as: *concentrating mass on activation maps*. If mass is not concentrated on individual activation maps prior to additive combination, then the resulting master map may be too nearly uniform and hence uninformative. Although this may seem trivial, it is on some level the very soul of any saliency algorithm: concentrating activation into a few key locations.

Armed with the mass-concentration definition, we propose another Markovian algorithm as follows: This time, we begin with an activation map[4] $A : [n]^2 \to \mathbb{R}$, which we wish to "normalize". We construct a graph $G_N$ with $n^2$ nodes labelled with indices from $[n]^2$. For each node $(i,j)$ and every node $(p,q)$ (including $(i,j)$) to which it is connected, we introduce an edge from $(i,j)$ to $(p,q)$ with weight:

$$w_2((i,j),(p,q)) \triangleq A(p,q) \cdot F(i-p,j-q).$$

Again, normalizing the weights of the outbound edges of each node to unity and treating the resulting graph as a Markov chain gives us the opportunity to compute the equilibrium distribution over the nodes[5]. Mass will flow preferentially to those nodes with high activation. It is a mass concentration algorithm by construction, and also one which is parallelizable, as before, having the same natural advantages. Experimentally, it seems to behave very favorably compared to the standard approaches such as "DoG" and "NL".

## 3 Experimental Results

### 3.1 Preliminaries and paradigm

We perform saliency computations on real images of the natural world, and compare the power of the resulting maps to predict human fixations. The experimental paradigm we pursue is the following: for each of a set of images, we compute a set of feature maps using standard techniques. Then, we proccess each of these feature maps using some activation algorithm, and then some normalization algorithm, and then simply sum over the feature channels. The resulting master saliency map is scored (using an ROC area metric described below) relative to fixation data collected for the corresponding image, and labelled according to the activation and normalization algorithms used to obtain it. We then pool over a corpus of images, and the resulting set of scored and labelled master saliency maps is analyzed in various ways presented below. Some notes follow:

**Algorithm Labels:** Hereafter, "graph (i)" and "graph (ii)" refer to the activation algorithm described in section 2.1.2. The difference is that in graph (i), the parameter $\sigma = 2.5$, whereas in graph (ii), $\sigma = 5$. "graph (iii)" and "graph (iv)" refer to the an iterated repitition of the normalization algorithm described in section 2.2. The difference is the termination rule associated with the iterative process: for graph (iii), a complicated termination rule is used which looks for a local maximum in the number of matrix multiplications required to achieve a stable equilibrium distribution[6], and for graph (iv), the termination rule is simply "stop after 4 iterations". The normalization algorithm referred to as "I" corresponds to "Identity", with the most naive normalization rule: it does nothing, leaving activations unchanged prior to subsequent combination. The algorithm "max-ave" and "DoG" were run using the publicly available "saliency toolbox"[7]. The parameters of this were checked against the literature [2] and [3], and were found to be almost identical, with a few slight alterations that actually improved performance relative to the published parameters. The parameters of "NL" were set according to the better of the two sets of parameters provided in [11].

**Performance metric:** We wish to give a reward quantity to a saliency map, given some target locations, e.g., in the case of natural images, a set of locations at which human observers fixated. For any one threshold saliency value, one can treat the saliency map as a classifier, with all points above threshold indicated as "target" and all points below threshold as "background". For any particular value of the threshold, there is some fraction of the actual target points which are labelled as such (true positive rate), and some fraction of points which were not target but labelled as such anyway (false positive rate). Varying over all such thresholds yields an ROC curve [14] and the area beneath it is generally regarded as an indication of the classifying power of the detector. This is the performance metric we use to measure how well a saliency map predicts fixation locations on a given image.

## 3.2 Human Eye-Movement Data on Images of Nature

In a study by Einhäuser et al. [1], human and primate fixation data was collected on 108 images, each modified[8] in nine ways. Figure 2 shows an example image from this collection, together with "x"s marking the fixation points of three human subjects on this particular picture. In the present study, 749 unique modifications of the 108 original images, and 24149 human fixations from [1] were used. Only pictures for which fixation data from three human subjects were available were used. Each image was cropped to $600 \times 400$ pixels and was presented to subjects so that it took up $76° \times 55°$ of their visual field. In order to facilitate a fair comparison of algorithms, the first step of the saliency algorithm, feature extraction (s1), was the same for every experiment. Two spatial scales $\left(\frac{1}{2}, \frac{1}{4}\right)$ were used, and for each of these, four orientation maps corresponding to orientations $\phi = \{0°, 45°, 90°, 135°\}$ were computed using Gabor filters, one contrast map was computed using luminance variance in a local neighborhood of size $80 \times 80$, and the last map was simply a luminance map (the grayscale values). Each of these 12 maps was finally downsampled to a $25 \times 37$ raw feature map.

"c-s" (center-surround) activation maps were computed by subtracting, from each raw feature map, a feature map on the same channel originally computed at a scale 4 binary orders of magnitude smaller in overall resolution and then resized smoothly to size $25 \times 37$. In [2], this overall scheme would be labelled $c = \{2, 3\}$, for $\frac{1}{2}$ and $\frac{1}{4}$, and $\delta = \{4\}$, corresponding to a scale change of 4 orders. The other activation procedures are described in section 2.1.2 and 2.1.1. The normalization procedures are all earlier described and named. Figure 2 shows an actual image with the resulting saliency maps from two different (activation, normalization) schemes.

(a) Sample Picture With Fixation

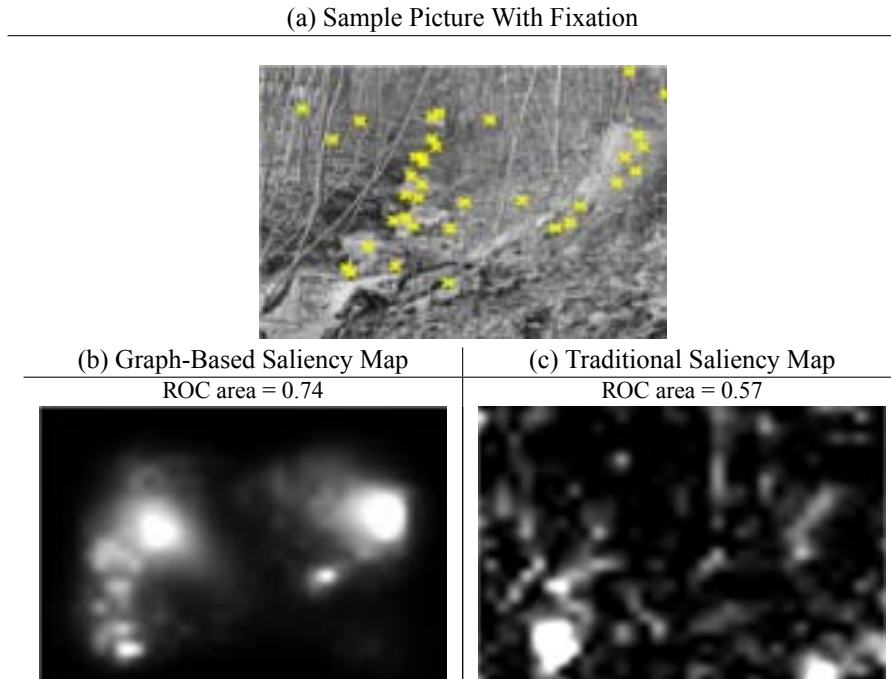

| (b) Graph-Based Saliency Map | (c) Traditional Saliency Map |
|---|---|
| ROC area = 0.74 | ROC area = 0.57 |

Figure 2: (a) An image from the data-set with fixations indicated using
x's. (b) The saliency map formed when using (activation,normalization)=
(graph (i),graph (iii)). (c) Saliency map for (activation,normalization)=(c-s,DoG)

Finally, we show the performance of this algorithm on the corpus of images. For each image, a mean inter-subject ROC area was computed as follows: for each of the three subjects who viewed an image, the fixation points of the remaining two subjects were convolved with a circular, decaying kernel with decay constant matched to the decaying cone density in the retina. This was treated as a saliency map derived directly from human fixations, and with the target points being set to the

fixations of the first subject, an ROC area was computed for a single subject. The mean over the three is termed "inter-subject ROC value" in the following figures. For each range of this quantity, a mean performance metric was computed for various activation and normalization schemes. For any particular scheme, an ROC area was computed using the resulting saliency map together with the fixations from all 3 human subjects as target points to detect. The results are shown below.

(a) Activation Comparison          (b) Normalization Comparison

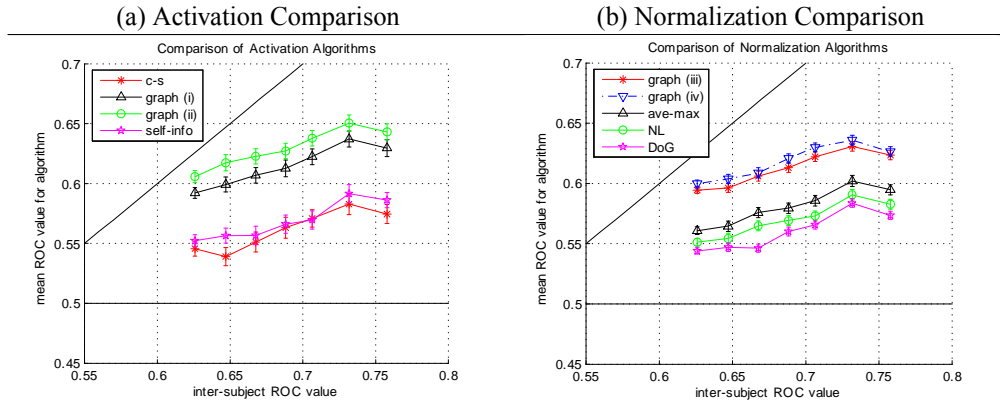

Figure 3: (a) A mean ROC metric is computed for each range of inter-subject ROC values. Each curve represents a different activation scheme, while averaging over individual image numbers and normalization schemes. (b) A mean ROC metric is similarly computed, instead holding the normalization constant while varying the activation scheme.

In both Figures 3 and 4, The boundary lines above and below show a rough upper[9] and strict lower bounds on performance (based on a human control and chance performance). Figure 3(a) and Figure 3(b) clearly demonstrate the tremendous predictive power of the graph-based algorithms over standard approaches. Figure 4 demonstrates the especially effective performance of combining the best graph-based activation and normalization schemes, contrasted against the standard Itti & Koch approaches, and also the "self-information" approach which includes no mention of a normalization step (hence, set here to "I").

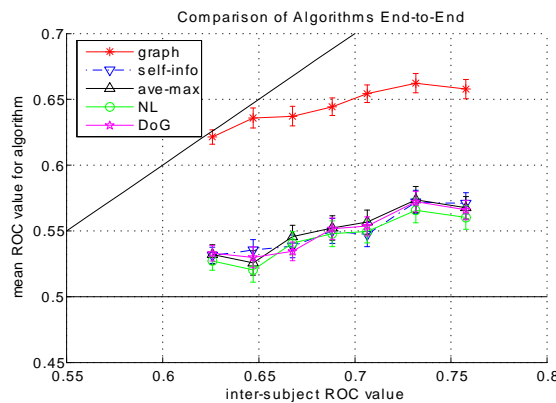

Figure 4: We compare the predictive power of five saliency algorithms. The best performer is the method which combines a graph based activation algorithm with a graph based normalization algorithm.

The combination of a few possible pairs of activation schemes together with normalization schemes is summarized in Table 1, with notes indicating where certain combinations correspond to established benchmarks. Performance is shown as a fraction of the inter-subject ROC area. Overall, we find an median ROC area of 0.55 for the Itti & Koch saliency algorithms [2] on these images. In [1]

the mean is reported as 0.57, which is remarkably close and plausible if you assume slightly more sophisticated feature maps (for instance, at more scales).

Table 1: Performance of end-to-end algorithms

| activation algorithm | normalization algorithm | ROC area (fraction[10]) | published |
|---|---|---|---|
| graph (ii) | graph (iv) | 0.981148 | |
| graph (i) | graph (iv) | 0.975313 | |
| graph (ii) | I | 0.974592 | |
| graph (ii) | ave-max | 0.974578 | |
| graph (ii) | graph (iii) | 0.974227 | |
| graph (i) | graph (iii) | 0.968414 | |
| self-info | I | 0.841054 | *Bruce & Tsotsos [5] |
| c-s | DoG | 0.840968 | *Itti & Koch [3] |
| c-s | ave-max | 0.840725 | *Itti, Koch, & Niebur [2] |
| c-s | NL | 0.831852 | *Lee, Itti, Koch, & Braun [10] |

## 4   Discussion and Conclusion

Although a novel, simple approach to an old problem is always welcome, we must also seek to answer the scientific question of how it is possible that, given access to the same feature information, GBVS predicts human fixations more reliably than the standard algorithms. We find experimentally that there are at least two reasons for this observed difference. The first observation is that, because nodes are on average closer to a few center nodes than to any particular point along the image periphery, it is an emergent property that GBVS promotes higher saliency values in the center of the image plane. We hypothesize that this "center bias" is favorable with respect to predicting fixations due to human experience both with photographs, which are typically taken with a central subject, and with everyday life in which head motion often results in gazing straight ahead. Notably, the images of foliage used in the present study *had no central subject*. One can quantify the GBVS-induced center bias by activating, then normalizing, a uniform image using our algorithms. However, if we introduce this center bias to the output of the standard algorithms' master maps (via point-wise multiplication), we find that the standard algorithms predict fixations better, but still worse than GBVS. In some cases (e.g., "DoG"), introducing this center bias only explains 20% of the performance gap to GBVS – in the best case (viz., "max-ave"), it explains 90% of the difference. We conjecture that the other reason for the performance difference stems from the robustness of our algorithm with respect to differences in the sizes of salient regions. Experimentally, we find that the "c-s" algorithm has trouble activating salient regions distant from object borders, even if one varies over many choices of scale differences and combinations thereof. Since most of the standard algorithms have "c-s" as a first step, they are weakened ab initio. Similarly, the "self-info" algorithm suffers the same weakness, even if one varies over the neighborhood size parameter. On the other hand, GBVS robustly highlights salient regions, even far away from object borders.

We note here that what lacks from GBVS described as above is any notion of a multiresolution representation of map data. Therefore, because multiresolution representations are so basic, one may extend both the graph-based activation and normalization steps to a multiresolution version as follows: We begin with, instead of a single map $A : [n]^2 \to \mathbb{R}$, a collection of maps $\{A_i\}$, with each $A_i : [n_i]^2 \to \mathbb{R}$ representing the same underlying information but at different resolutions. Proceeding as we did before, we instantiate a node for every point *on every map*, introducing edges again between every pair of nodes, with weights computed same as before with one caveat: the

distance penalty function $F(a, b)$ accepts two arguments each of which is a distance between two nodes along a particular dimension. In order to compute $F$ in this case, one must define a distance over points taken from different underlying domains. The authors suggest a definition whereby: (1) each point in each map is assigned a set of locations, (2) this set corresponds to the spatial support of this point in the highest resolution map, and (3) the distance between two sets of locations is given as the mean of the set of pairwise distances. The equilibrium distribution can be computed as before. We find that this extension (say, GBVS Multiresolution, or GBVSM) improves performance with little added computation.

Therefore, we have presented a method of computing bottom-up saliency maps which shows a remarkable consistency with the attentional deployment of human subjects. The method uses a novel application of ideas from graph theory to concentrate mass on activation maps, and to form activation maps from raw features. We compared our method with established models and found that ours performed favorably, for both of the key steps in our organization of saliency computations. Our model is extensible to multiresolutions for better performance, and it is biologically plausible to the extent that a parallel implementation of the power-law algorithm for Markov chains is trivially accomplished in hardware.

## Acknowledgments

The authors express sincere gratitude to Wolfgang Einhäuser for his offering of natural images, and the fixation data associated with them from a study with seven human subjects. We also acknowledge NSF, NIH, DARPA, and ONR for their generous support of our research.

## Footnotes

[1]in the context of a mathematical formulation, let $[n] \triangleq \{1, 2, ..., n\}$. Also, the maps $M$, and later $A$, are presented as square ($n \times n$) only for expository simplicity. Nothing in this paper will depend critically on the square assumtion, and, in practice, rectangular maps are used instead.

[2]In our experiments, this parameter was set to approximately one tenth to one fifth of the map width. Results were not very sensitive to perturbations around these values.

[3]Our implementation, not optimized for speed, converges on a single map of size $25 \times 37$ in fractions of a second on a 2.4 GHz Pentium.

[4]To be clear, if $A$ is the result of the eigenvector computation described in 2.1, i.e., if the graph-based activation step is concatenated with the graph-based normalization step, we will call the resulting algorithm GBVS. However, $A$ may be computed using other techniques.

[5]We note that this normalization step of GBS can be iterated $\kappa$ times to improve performance. In practice, we use $\kappa \in \{2,3,4\}$. Performance does not vary significantly in this regime with respect to $\kappa$.

[6]with the intuition being that competition among competing saliency regions can settle, at which point it is wise to terminate

[7]http://www.saliencytoolbox.net

[8]Modifications were made to change the luminance contrast either up or down in selected circular regions. Both modified and unmodified stimuli were used in these experiments. Please refer to [1], [12].

[9]To form a true upper bound, one would need the fixation data of many more than three humans on each image.

[10]performance here is measured by the ratio of (ROC area using the given algorithm for fixation detection) to (ROC area using a saliency map formed from the fixations of *other* subjects on a single picture)

## References

[1] W. Einhäuser, W. Kruse, K.P. Hoffmann, & P. König "Differences of Monkey and Human Overt Attention under Natural Conditions", *Vision Research* 2006.

[2] L. Itti, C. Koch, & E. Niebur "A model of saliency based visual attention for rapid scene analysis", *IEEE Transactions on Pattern Analysis and Machine 1998*

[3] L. Itti & C. Koch "A saliency-based search mechanism for overt and covert shifts of visual attention", *Vision Research*, 2000

[4] L. Itti, & P. Baldi "Bayesian Surprise Attracts Human Attention", *NIPS*2005*

[5] N. Bruce & J. Tsotsos "Saliency Based on Information Maximization", *NIPS*2005*

[6] L.F. Costa "Visual Saliency and Attention as Random Walks on Complex Networks", *arXiv preprint* 2006

[7] G. Boccignone, & M. Ferraro "Modelling gaze shift as a constrained random walk", *Physica A 331, 207* 2004

[8] D. Brockmann, T. Geisel "Are human scanpaths Levy flights?", *ICANN 1999*

[9] D. Parkhurst, K. Law, & E. Niebur "Modeling the role of salience in the allocation of overt visual attention", *Vision Research*, 2002

[10] D.K. Lee, L. Itti, C. Koch, & J. Braun "Attention activates winner-take-all competition among visual features", *Nature Neuroscience*, 1999

[11] L. Itti, J. Braun, D.K. Lee, & C. Koch "Attention Modulation of Human Pattern Discrimination Psychophysics Reproduced by a Quantitative Model", *NIPS*1998*

[12] W. Einhäuser & P. König, "Does luminance-contrast contribute to saliency map for overt visual attention?", *Eur. J. Neurosci.* 2003

[13] U. Rutishauser, D. Walther, C. Koch, & P. Perona "Is bottom-up attention useful for object recognition?", *CVPR 2004*

[14] B.W. Tatler, R.J. Baddeley, & I.D. Gilchrist "Visual correlates of fixation selection: Effects of scale and time." *Vision Research* 2005

[15] J. Malik & P. Perona "Preattentive texture discrimination with early vision mechanisms" *Journal of the Optical Society of America A* 1990
